# Spike Sorting: Bayesian Clustering of Non-Stationary Data

**Aharon Bar-Hillel**
Neural Computation Center
The Hebrew University of Jerusalem
`aharonbh@cs.huji.ac.il`

**Adam Spiro**
School of Computer Science and Engineering
The Hebrew University of Jerusalem
`adams@cs.huji.ac.il`

**Eran Stark**
Department of Physiology
The Hebrew University of Jerusalem
`eranstark@md.huji.ac.il`

## Abstract

Spike sorting involves clustering spike trains recorded by a micro-electrode according to the source neuron. It is a complicated problem, which requires a lot of human labor, partly due to the non-stationary nature of the data. We propose an automated technique for the clustering of non-stationary Gaussian sources in a Bayesian framework. At a first search stage, data is divided into short time frames and candidate descriptions of the data as a mixture of Gaussians are computed for each frame. At a second stage transition probabilities between candidate mixtures are computed, and a globally optimal clustering is found as the MAP solution of the resulting probabilistic model. Transition probabilities are computed using local stationarity assumptions and are based on a Gaussian version of the Jensen-Shannon divergence. The method was applied to several recordings. The performance appeared almost indistinguishable from humans in a wide range of scenarios, including movement, merges, and splits of clusters.

## 1 Introduction

Neural spike activity is recorded with a micro-electrode which normally picks up the activity of multiple neurons. Spike sorting seeks the segmentation of the spike data such that each cluster contains all the spikes generated by a different neuron. Currently, this task is mostly done manually. It is a tedious mission, requiring many hours of human labor for each recording session. Several algorithms were proposed in order to help automating this process (see [7] for a review, [9],[10]) and some tools were implemented to assist in manual sorting [8]. However, the ability of suggested algorithms to replace the human worker has been quite limited.

One of the main obstacles to a successful application is the non-stationary nature of the data [7]. The primary source of this non-stationarity is slight movements of the recording elec-

trode. Slight drifts of the electrode's location, which are almost inevitable, cause changes in the typical shapes of recorded spikes over time. Other sources of non-stationarity include variable background noise and changes in the characteristic spike generated by a certain neuron. The increasing usage of multiple electrode systems turns non-stationarity into an acute problem, as electrodes are placed in a single location for long durations.

Using the first 2 PCA coefficients to represent the data (which preserves up to $93\%$ of the variance in the original recordings [1]), a human can cluster spikes by visual inspection. When dividing the data into small enough time frames, cluster density can be approximated by a multivariate Gaussian with a general covariance matrix without loosing much accuracy [7]. Problematic scenarios which can appear due to non-stationarity are exemplified in Section 4.2 and include: (1) Movements and considerable shape changes of the clusters over time, (2) Two clusters which are reasonably well-separated may move until they converge and become indistinguishable. A split of a cluster is possible in the same manner.

Most spike sorting algorithms do not address the presented difficulties at all, as they assume full stationarity of the data. Some methods [4, 11] try to cope with the lack of stationarity by grouping data into many small clusters and identifying the clusters that can be combined to represent the activity of a single unit. In the second stage, [4] uses ISI information to understand which clusters cannot be combined, while [11] bases this decision on the density of points between clusters. In [3] a semi-automated method is suggested, in which each time frame is clustered manually, and then the correspondence between clusters in consecutive time frames is established automatically. The correspondence is determined by a heuristic score, and the algorithm doesn't handle merge or split scenarios.

In this paper we suggest a new fully automated technique to solve the clustering problem for non-stationary Gaussian sources in a Bayesian framework. We divide the data into short time frames in which stationarity is a reasonable assumption. We then look for good mixture of Gaussians descriptions of the data in each time frame independently. Transition probabilities between local mixture solutions are introduced, and a globally optimal clustering solution is computed by finding the Maximum-A-Posteriori (MAP) solution of the resulting probabilistic model. The global optimization allows the algorithm to successfully disambiguate problematic time frames and exhibit close to human performance. We present the outline of the algorithm in Section 2. The transition probabilities are computed by optimizing the Jensen-Shannon divergence for Gaussians, as described in Section 3. Empirical results and validation are presented in Section 4.

## 2  Clustering using a chain of Gaussian mixtures

Denote the observable spike data by $D = \{d\}$, where each spike $d \in R^n$ is described by the vector of its PCA coefficients. We break the data into $T$ disjoint groups $\{D_t = \{d_i^t\}_{i=1}^{N_t}\}_{t=1}^{T}$. We assume that in each frame, the data can be well approximated by a mixture of Gaussians, where each Gaussian corresponds to a single neuron. Each Gaussian in the mixture may have a different covariance matrix. The number of components in the mixture is not known a priori, but is assumed to be within a certain range (we used 1-6).

In the search stage, we use a standard EM (Expectation-Maximization) algorithm to find a set of $M^t$ candidate mixture descriptions for each time frame $t$. We build the set of candidates using a three step process. First, we run the EM algorithm with different number of clusters and different initial conditions. In a second step, we import to each time frame $t$ the best mixture solutions found in the neighboring time frames $[t - k, .., t + k]$ (we used $k = 2$). These solutions are also adapted by using them as the initial conditions for the EM and running a low number of EM rounds. This mixing of solutions between time frames is repeated several times. Finally, the solution list in each time frame is pruned to remove similar solutions. Solutions which don't comply with the assumption of well

shaped Gaussians are also removed.

In order to handle outliers, which are usually background spikes or non-spike events, each mixture candidate contains an additional 'background model' Gaussian. This model's parameters are set to $0, K \cdot \Sigma_t$ where $\Sigma_t$ is the covariance matrix of the data in frame $t$ and $K > 1$ is a constant. Only the weight of this model is allowed to change during the EM process.

After the search stage, each time frame $t$ has a list of $M^t$ models $\{\Theta_i^t\}_{t=1,i=1}^{T,M^t}$. Each mixture model is described by a triplet $\Theta_i^t = \{\alpha_{i,l}^t, \mu_{i,l}^t, \Sigma_{i,l}^t\}_{l=1}^{K_{i,t}}$, denoting Gaussian mixture's weights, means, and covariances respectively. Given these candidate models we define a discrete random vector $Z = \{z^t\}_{t=1}^T$ in which each component $z^t$ has a value range of $\{1, 2, .., M^t\}$. "$z^t = j$" has the semantics of "at time frame $t$ the data is distributed according to the candidate mixture $\Theta_j^t$". In addition we define for each spike $d_i^t$ a hidden discrete 'label' random variable $l_i^t$. This label indicates which Gaussian in the local mixture hypothesis is the source of the spike. Denote by $L^t = \{l_i^t\}_{i=1}^{N^t}$ the vector of labels of time frame $t$, and by $L$ the vector of all the labels.

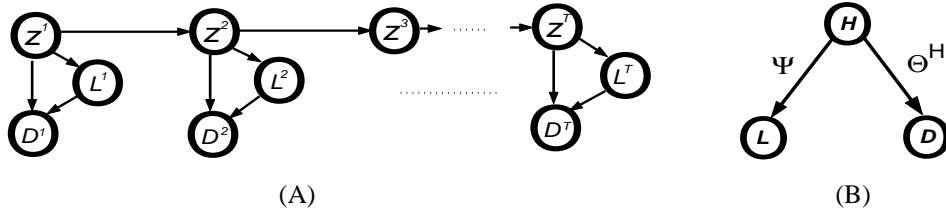

<div align="center">(A)          (B)</div>

Figure 1: (A) A Bayesian network model of the data generation process. The network has an HMM structure, but unlike HMM it does not have fixed states and transition probabilities over time. The variables and the CPDs are explained in Section 2. (B) A Bayesian network representation of the relations between the data $D$ and the hidden labels $H$ (see Section 3.1). The visible labels $L$ and the sampled data points are independent given the hidden labels.

We describe the probabilistic relations between $D$, $L$, and $Z$ using a Bayesian network with the structure described in Figure 1A. Using the network structure and assuming i.i.d samples the joint log probability decomposes into

$$\log P(z^1) + \sum_{t=2}^{T} \log P(z^t|z^{t-1}) + \sum_{t=1}^{T}\sum_{i=1}^{N^t} [\log P(l_i^t|z^t) + \log P(d_i^t|l_i^t, z^t)] \quad (1)$$

We wish to maximize this log-likelihood over all possible choices of $L, Z$. Notice that by maximizing the probability of both data and labels we avoid the tendency to prefer mixtures with many Gaussians, which appears when maximizing the probability for the data alone. The conditional probability distributions (CPDs) of the points' labels and the points themselves, given an assignment to $Z$, are given by

$$\log P(l_k^t = j|z^t = i) = \log \alpha_{i,j}^t \quad (2)$$

$$\log P(d_k^t|l_i^t = j, z^t = i) = -\frac{1}{2}[n \log 2\pi + \log |\Sigma_{i,j}^t| + (d_k^t - \mu_{i,j}^t)^t \Sigma_{i,j}^{t}{}^{-1}(d_k^t - \mu_{i,j}^t)]$$

The transition CPDs $P(z^t|z^{t-1})$ are described in Section 3. For the first frame's prior we use a uniform CPD. The MAP solution for the model is found using the Viterbi algorithm. Labels are then unified using the correspondences established between the chosen mixtures in consecutive time frames. As a final adjustment step, we repeat the mixing process using only the mixtures of the found MAP solution. Using this set of new candidates, we calculate the final MAP solution in the same manner described above.

# 3 A statistical distance between mixtures

The transition CPDs of the form $P(z^t|z^{t-1})$ are based on the assumption that the Gaussian sources' distributions are approximately stationary in pairs of consecutive time frames. Under this assumption, two mixtures candidates estimated at consecutive time frames are viewed as two samples from a single unknown Gaussian mixture. We assume that each Gaussian component from any of the two mixtures arises from a single Gaussian component in the joint hidden mixture, and so the hidden mixture induces a partition of the set of visible components into clusters. Gaussian components in the same cluster are assumed to arise from the same hidden source. Our estimate of $p(z^t = j|z^{t-1} = i)$ is based on the probability of seeing two large samples with different empirical distributions ($\Theta_i^{t-1}$ and $\Theta_j^t$ respectively) under the assumption of such a single joint mixture. In Section 3.1, the estimation of the transition probability is formalized as an optimization of a Jensen-Shannon based score over the possible partitions of the Gaussian components set.

If the family of allowed hidden mixture models is not further constrained, the optimization problem derived in Section 3.1 is trivially solved by choosing the most detailed partition (each visible Gaussian component is a singleton). This happens because a richer partition, which does not merge many Gaussians, gets a higher score. In Section 3.2 we suggest natural constraints on the family of allowed partitions in the two cases of constant and variable number of clusters through time, and present algorithms for both cases.

## 3.1 A Jensen-Shannon based transition score

Assume that in two consecutive time frames we observed two labeled samples $(X^1, L^1), (X^2, L^2)$ of sizes $N^1, N^2$ with empirical distributions $\Theta^1, \Theta^2$ respectively. By 'empirical distribution', or 'type' in the notation of [2], we denote the ML parameters of the sample, for both the multinomial distribution of the mixture weights and the Gaussian distributions of the components. As stated above, we assume that the joint sample of size $N = N^1 + N^2$ is generated by a hidden Gaussian mixture $\Theta^H$ with $K^H$ components, and its components are determined by a partition of the set of all components in $\Theta^1, \Theta^2$. For convenience of notation, let us order this set of $K^1 + K^2$ Gaussians and refer to them (and to their parameters) using one index. We can define a function $R : \{1, .., K^1 + K^2\} \rightarrow \{1, .., K^H\}$ which matches each visible Gaussian component in $\Theta^1$ or $\Theta^2$ to its hidden source component in $\Theta^H$. Denote the labels of the sample points under the hidden mixture $H = \{h_i^j\}_{i=1}^{N^j}, \quad j = 1, 2$. The values of these variables are given by $h_i^j = R(l_i^j)$, where $l_i^j$ is the label index in the set of all components.

The probabilistic dependence between a data point, its visible label, and its hidden label is explained by the Bayesian network model in Figure 1B. We assume a data point is obtained by choosing a hidden label and then sample the point from the relevant hidden component. The visible label is then sampled based on the hidden label using a multinomial distribution with parameters $\Psi = \{\Psi_q\}_{q=1}^{K^1+K^2}$, where $\Psi_q = P(l = q|h = R(q))$, i.e., the probability of the visible label $q$ given the hidden label $R(q)$ (since $H$ is deterministic given $L$, $P(l = q|h) = 0$ for $h \neq R(q)$). Denote this model, which is fully determined by $R, \Psi$, and $\Theta^H$, by $M^H$.

We wish to estimate $P((X^1, L^1) \sim \Theta^1|(X^2, L^2) \sim \Theta^2, M^H)$. We use ML approximations and arguments based on the method of types [2] to approximate this probability and optimize it with respect to $\Theta^H$ and $\Psi$. The obtained result is (the derivation is omitted)

$$P((X^1, L^1) \sim \Theta^1|(X^2, L^2) \sim \Theta^2, M^H) \approx \qquad (3)$$

$$\max_R \exp(-N \cdot \sum_{m=1}^{K^H} \alpha_m^H \sum_{\{q:R(q)=m\}} \Psi_q D_{kl}(G(x|\mu_q, \Sigma_q)|G(x|\mu_m^H, \Sigma_m^H)))$$

where $G(x|\mu, \Sigma)$ denotes a Gaussian distribution with the parameters $\mu, \Sigma$ and the optimized $\Theta^H, \Psi$ appearing here are given as follows. Denote by $w_q$ ($q \in \{1,.., K^1 + K^2\}$) the weight of model $q$ in a naive joint mixture of $\Theta^1, \Theta^2$, i.e., $w_q = \frac{N^j}{N} \alpha_q$ where $j = 1$ if component $q$ is part of $\Theta^1$ and the same for $j = 2$.

$$\alpha_m^H = \sum_{\{q:R(q)=m\}} w_q \quad , \quad \Psi_q = \frac{w_q}{\alpha_{R(q)}^H} \quad , \quad \mu_m^H = \sum_{\{q:R(q)=m\}} \Psi_q \mu_q \quad (4)$$

$$\Sigma_m^H = \sum_{\{q:R(q)=m\}} \Psi_q(\Sigma_q + (\mu_q - \mu_m^H)(\mu_q - \mu_m^H)^t)$$

Notice that the parameters of a hidden Gaussian, $\mu_m^H$ and $\Sigma_m^H$, are just the mean and covariance of the mixture $\sum_{q:R(q)=m} \Psi_q G(x|\mu_q, \Sigma_q)$. The summation over $q$ in expression (3) can be interpreted as the Jensen-Shannon (JS) divergence between the components assigned to the hidden source $m$, under Gaussian assumptions.

For a given parametric family, the JS divergence is a non-negative measurement which can be used to test whether several samples are derived from a single distribution from the family or from a mixture of different ones [6]. The JS divergence is computed for a mixture of $n$ empirical distributions $P_1,.., P_n$ with mixture weights $\pi_1,.., \pi_n$. In the Gaussian case, denote the mean and covariance of the component distributions by $\{\mu_i, \Sigma_i\}_{i=1}^n$. The mean and covariance of the mixture distribution $\mu^*, \Sigma^*$ are a function of the means and covariances of the components, with the formulae given in (4) for $\mu_m^H, \Sigma_m^H$. The Gaussian JS divergence is given by

$$JS_{\pi_1,..,\pi_n}^G(P_1,.., P_n) = \sum_{i=1}^n \pi_i D_{kl}(G(x|\mu_i, \Sigma_i), G(x|\mu^*, \Sigma^*)) \quad (5)$$

$$= H(G(x|\mu^*, \Sigma^*)) - \sum_{i=1}^n \pi_i H(G(x|\mu_i, \Sigma_i)) = \frac{1}{2}(\log|\Sigma^*| - \sum_{i=1}^n \pi_i \log|\Sigma_i|)$$

using this identity in (3), and setting $\Theta^1 = \Theta_i^t, \Theta^2 = \Theta_j^{t-1}$, we finally get the following expression for the transition probability

$$\log P(z^t = i|z^{t-1} = j) = \quad (6)$$

$$-N \cdot \max_R \sum_{m=1}^{K^H} \alpha_m^H JS_{\{\Psi_q:R(q)=m\}}^G(\{G(x|\mu_q, \Sigma_q) : R(q) = m\})$$

### 3.2 Constrained optimization and algorithms

Consider first the case in which a one-to-one correspondence is assumed between clusters in two consecutive frames, and hence the number of Gaussian components $K$ is constant over all time frames. In this case, a mapping $R$ is allowed iff it maps to each hidden source $i$ a single Gaussian from mixture $\Theta^1$ and a single Gaussian from $\Theta^2$. Denoting the Gaussians matched to hidden $i$ by $R_1^{-1}(i), R_2^{-1}(i)$, the transition score (6) takes the form of $-N \cdot \max_R \sum_{i=1}^K S(R_1^{-1}(i), R_2^{-1}(i))$. Such an optimization of a pairwise matching score can be seen as a search for a maximal perfect matching in a weighted bipartite graph. The nodes of the graph are the Gaussian components of $\Theta^1, \Theta^2$ and the edges' weights are

given by the scores $S(a, b)$. The global optimum of this problem can be efficiently found using the Hungarian algorithm [5] in $O(n^3)$, which is unproblematic in our case.

The one-to-one correspondence assumption is too strong for many data sets in the spike sorting application, as it ignores the phenomena of splits and merges of clusters. We wish to allow such phenomena, but nevertheless enforce strong (though not perfect) demands of correspondence between the Gaussians in two consecutive frames. In order to achieve such balance, we place the following constraints on the allowed partitions $R$:

1. Each cluster of $R$ should contain exactly one Gaussian from $\Theta^1$ or exactly one Gaussian from $\Theta^2$. Hence assignment of different Gaussians from the same mixture to the same hidden source is limited only for cases of a split or a merge.

2. The label entropy of the partition $R$ should satisfy

$$H(\alpha_1^H, .., \alpha_{K^H}^H) \leq \frac{N^1}{N} H(\alpha_1^1, .., \alpha_{K^1}^1) + \frac{N^2}{N} H(\alpha_1^2, .., \alpha_{K^2}^2) \qquad (7)$$

Intuitively, the second constraint limits the allowed partitions to ones which are not richer than the visible partition, i.e., do not have much more clusters. Note that the most detailed partition (the partition into singletons) has a label entropy given by the r.h.s of inequality (7) plus $H(\frac{N^1}{N}, \frac{N^2}{N})$, which is one bit for $N^1 = N^2$. This extra bit is the price of using the concatenated 'rich' mixture, so we look for mixtures which do not pay such an extra price.

The optimization for this family of $R$ does not seem to have an efficient global optimization technique, and thus we resort to a greedy procedure. Specifically, we use a bottom up agglomerative algorithm. We start from the most detailed partition (each Gaussian is a singleton) and merge two clusters of the partition at each round. Only merges that comply with the first constraint are considered. At each round we look for a merge which incurs a minimal loss to the accumulated Jensen Shannon score (6) and a maximal loss to the mixture label entropy. For two Gaussian clusters $(\alpha_1, \mu_1, \Sigma_1)$, $(\alpha_2, \mu_2, \Sigma_2)$ these two quantities are given by

$$\Delta \log JS = -N(w_1 + w_2) JS_{\pi_1, \pi_2}^G (G(x|\mu_1, \Sigma_1), G(x|\mu_2, \Sigma_2)) \qquad (8)$$
$$\Delta H = -N(w_1 + w_2) H(\pi_1, \pi_2)$$

where $\pi_1, \pi_2$ are $\frac{w_1}{w_1 + w_2}, \frac{w_2}{w_1 + w_2}$ and $w_i$ are as in (4). We choose at each round the merge which minimizes the ratio between these two quantities. The algorithm terminates when the accumulated label entropy reduction is bigger than $H(\frac{N^1}{N}, \frac{N^2}{N})$ or when no allowed merges exist anymore. In the second case, it may happen that the partition $R$ found by the algorithm violates the constraint (7). We nevertheless compute the score based on the $R$ found, since this partition obeys the first constraint and usually is not far from satisfying the second.

## 4 Empirical results

### 4.1 Experimental design and data acquisition

Neural data were acquired from the dorsal and ventral pre-motor (PMd, PMv) cortices of two Macaque monkeys performing a prehension (reaching and grasping) task. At the beginning of each trial, an object was presented in one of six locations. Following a delay period, a Go signal prompted the monkey to reach for, grasp, and hold the target object. A recording session typically lasted 2 hours during which monkeys completed 600 trials. During each session 16 independently-movable glass-plated tungsten micro-electrodes

| $f_{\frac{1}{2}}$ score | Number of frames (%) | Number of electrodes (%) |
|---|---|---|
| 0.9-1.0 | 3386 (75%) | 13 (30%) |
| 0.8-0.9 | 860 (19%) | 10 (23%) |
| 0.7-0.8 | 243 (5%) | 10 (23%) |
| 0.6-0.7 | 55 (1%) | 11 (25%) |

Table 1: Match scores between manual and automatic clustering. The rows list the appearance frequencies of different $f_{\frac{1}{2}}$ scores.

were inserted through the dura, 8 into each area. Signals from these electrodes were amplified (10K), bandpass filtered (5-6000Hz), sampled (25 kHz), stored on disk (Alpha-Map 5.4, Alpha-Omega Eng.), and subjected to 3-stage preprocessing. (1) Line influences were cleaned by pulse-triggered averaging: the signal following a pulse was averaged over many pulses and subtracted from the original in an adaptive manner. (2) Spikes were detected by a modified second derivative algorithm (7 samples backwards and 11 forward), accentuating spiky features; segments that crossed an adaptive threshold were identified. Within each segment, a potential spike's peak was defined as the time of the maximal derivative. If a sharper spike was not encountered within 1.2ms, 64 samples (10 before peak and 53 after) were registered. (3) Waveforms were re-aligned s.t. each started at the point of maximal fit with 2 library PCs (accounting, on average, for $82\%$ and $11\%$ of the variance, [1]). Aligned waveforms were projected onto the PCA basis to arrive at two coefficients.

## 4.2 Results and validation

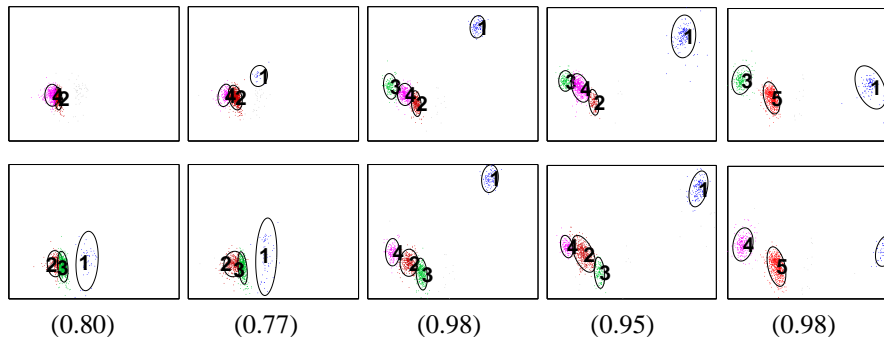

(0.80)  (0.77)  (0.98)  (0.95)  (0.98)

Figure 2: Frames 3,12,24,34, and 47 from a 68-frames data set. Each frame contains 1000 spikes, plotted here (with random number assignments) according to their first two PCs. In this data one cluster moves constantly, another splits into distinguished clusters, and at the end two clusters are merged. The top and bottom rows show manual and automatic clustering solutions respectively. Notice that during the split process of the bottom left area some ambiguous time frames exist in which 1,2, or 3 cluster descriptions are reasonable. This ambiguity can be resolved using global considerations of past and future time frames. By finding the MAP solution over all time frames, the algorithm manages such considerations. The numbers below the images show the $f_{\frac{1}{2}}$ score of the local match between the manual and the automatic clustering solutions (see text).

We tested the algorithm using recordings of 44 electrodes containing a total of 4544 time frames. Spike trains were manually clustered by a skilled user in the environment of Alpha-Sort 4.0 (Alpha-Omega Eng.). The manual and automatic clustering results were compared using a combined measure of precision $P$ and recall $R$ scores $f_{\frac{1}{2}} = \frac{2PR}{R+P}$. Figure 2 demonstrates the performance of the algorithm using a particularly non-stationary data set.

Statistics on the match between manual and automated clustering are described in Table 1. In order to understand the score's scale we note that random clustering (with the same

label distribution as the manual clustering) gets an $f_{\frac{1}{2}}$ score of $0.5$. The trivial clustering which assigns all the points to the same label gets mean scores of $0.73$ and $0.67$ for single frame matching and whole electrode matching respectively. The scores of single frames are much higher than the full electrode scores, since the problem is much harder in the latter case. A single wrong correspondence between two consecutive frames may reduce the electrode's score dramatically, while being unnoticed by the single frame score. In most cases the algorithm gives reasonably evolving clustering, even when it disagrees with the manual solution. Examples can be seen at the authors' web site[1].

Low matching scores between the manual and the automatic clustering may result from inherent ambiguity in the data. As a preliminary assessment of this hypothesis we obtained a second, independent, manual clustering for the data set for which we got the lowest match scores. The matching scores between manual and automatic clustering are presented in Figure 3A.

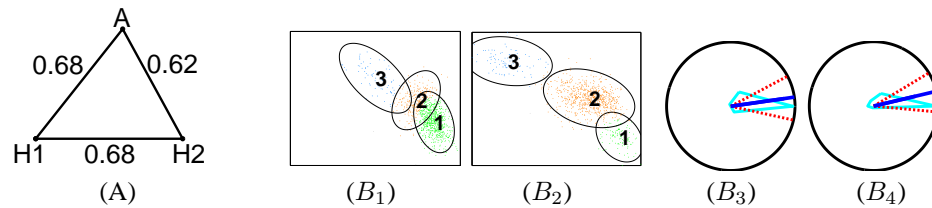

Figure 3: (A) Comparison of our automatic clustering with 2 independent manual clustering solutions for our worst matched data points. Note that there is also a low match between the humans, forming a nearly equilateral triangle. (B) Functional validation of clustering results: (1) At the beginning of a recording session, three clusters were identified. (2) 107 minutes later, some shifted their position. They were tracked continuously. (3) The directional tuning of the top left cluster (number 3) during the delay periods of the first 100 trials (dashed lines are $99\%$ confidence limits). (4) Although the cluster's position changed, its tuning curve's characteristics during the last 100 trials were similar.

In some cases, validity of the automatic clustering can be assessed by checking functional properties associated with the underlying neurons. In Figure 3B we present such a validation for a successfully tracked cluster.

## Footnotes

[1]http://www.cs.huji.ac.il/~aharonbh,~adams

# References

[1] Abeles M., Goldstein M.H. Multispike train analysis. Proc IEEE 65, pp. 762-773, 1977.

[2] Cover T., Thomas J. Elements of information theory. John wiley and sons, New York 1991.

[3] Emondi A.A, Rebrik S.P, Kurgansky A.V, Miller K.D. Tracking neurons recorded from tetrodes across time. J. of Neuroscience Methods, vol. 135:95-105, 2004.

[4] Fee M., Mitra P., Kleinfeld D. Automatic sorting of multiple unit neuronal signals in the presence of anisotropic and non-gaussian variability. J. of Neuroscience Methods, vol. 69:175-188, 1996.

[5] Kuhn H.W. The Hungarian method for the assignment problem. Naval research logistics quarterly, pp. 83-87, 1995.

[6] Lehmann E.L. Testing statistical hypotheses John Wiley and Sons, New York 1959.

[7] Lewicki, M.S. A review of methods for spike sorting: the detection and classification of neural action potentials. Network: Computation in Neural Systems. 9(4):R53-R78, 1998.

[8] Lewicki's Bayesian spike sorter, sslib (ftp.etho.caltech.edu).

[9] Penev P., Dimitrov A., Miller J. Characterization of and compensation for the non-stationarity of spike shapes during physiological recordings. Neurocomputing 38-40:1695-1701, 2001.

[10] Shoham S., Fellows M.R., Normann R.A. Robust, automatic spike sorting using mixtures of multivariate t-distributions. J. of Neuroscience Methods vol. 127(2):111-122, 2003.

[11] Snider R.K. , Bonds A.B. Classification of non-stationary neural signals. J. of Neuroscience Methods, vol. 84(1-2):155-166, 1998.

